# Modeling Short-term Noise Dependence of Spike Counts in Macaque Prefrontal Cortex

**Arno Onken**
Technische Universität Berlin
/ BCCN Berlin
aonken@cs.tu-berlin.de

**Steffen Grünewälder**
Technische Universität Berlin
Franklinstr. 28/29, 10587 Berlin, Germany
gruenew@cs.tu-berlin.de

**Matthias Munk**
MPI for Biological Cybernetics
Spemannstr. 38, 72076 Tübingen, Germany
matthias.munk@tuebingen.mpg.de

**Klaus Obermayer**
Technische Universität Berlin
/ BCCN Berlin
oby@cs.tu-berlin.de

## Abstract

Correlations between spike counts are often used to analyze neural coding. The noise is typically assumed to be Gaussian. Yet, this assumption is often inappropriate, especially for low spike counts. In this study, we present copulas as an alternative approach. With copulas it is possible to use arbitrary marginal distributions such as Poisson or negative binomial that are better suited for modeling noise distributions of spike counts. Furthermore, copulas place a wide range of dependence structures at the disposal and can be used to analyze higher order interactions. We develop a framework to analyze spike count data by means of copulas. Methods for parameter inference based on maximum likelihood estimates and for computation of mutual information are provided. We apply the method to our data recorded from macaque prefrontal cortex. The data analysis leads to three findings: (1) copula-based distributions provide significantly better fits than discretized multivariate normal distributions; (2) negative binomial margins fit the data significantly better than Poisson margins; and (3) the dependence structure carries 12% of the mutual information between stimuli and responses.

## 1 Introduction

Understanding neural coding is at the heart of theoretical neuroscience. Analyzing spike counts of a population is one way to gain insight into neural coding properties. Even when the same stimulus is presented repeatedly, responses from the neurons vary, i.e. from trial to trial responses of neurons are subject to noise. The noise variations of neighboring neurons are typically correlated (*noise correlations*). Due to their relevance for neural coding, noise correlations have been subject of a considerable number of studies (see [1] for a review). However, these studies always assumed Gaussian noise. Thus, correlated spike rates were generally modeled by multivariate normal distributions with a specific covariance matrix that describes all pairwise linear correlations.

For long time intervals or high firing rates, the average number of spikes is sufficiently large for the *central limit theorem* to apply and thus the normal distribution is a good approximation for the spike count distributions. However, several experimental findings suggest that noise correlations as well as sensory information processing predominantly take place on a shorter time scale, on the order of tens to hundreds of milliseconds [2, 3]. It is therefore questionable if the normal distribution is still an appropriate approximation and if the results of studies based on Gaussian noise apply to short time intervals and low firing rates.

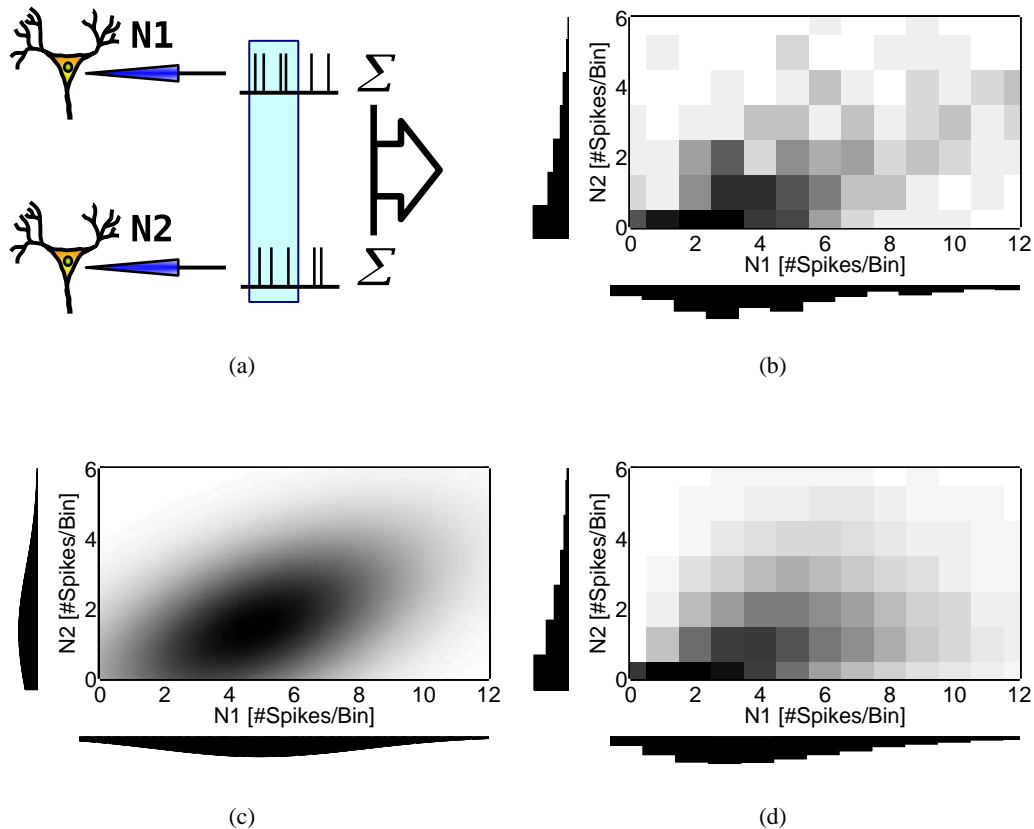

Figure 1: (**a**): Recording of correlated spike trains from two neurons and conversion to spike counts. (**b**): The distributions of the spike counts of a neuron pair from the data described in Section 4 for 100 ms time bins. Dark squares represent a high number of occurrences of corresponding pairs of spike counts. One can see that the spike counts are correlated since the ratios are high near the diagonal. The distributions of the individual spike counts are plotted below and left of the axes. (**c**): Density of a fit with a bivariate normal distribution. (**d**): Distribution of a fit with negative binomial margins coupled with the Clayton copula.

This is due to several major drawbacks of the multivariate normal distribution: (1) Its margins are continuous with a symmetric shape, whereas empirical distributions of real spike counts tend to have a positive skew, i.e. the mass of the distribution is concentrated at the left of its mode. Moreover, the normal distribution allows negative values which are not meaningful for spike counts. Especially for low rates, this can become a major issue, since the probability of negative values will be high. (2) The dependence structure of a multivariate normal distribution is always elliptical, whereas spike counts of short time bins can have a bulb-shaped dependence structure (see Fig. 1**b**). (3) The multivariate normal distribution does not allow higher order correlations of its elements. Instead, only pairwise correlations can be modeled. It was shown that pairwise interactions are sufficient for retinal ganglion cells and cortex cells *in vitro* [4]. However, there is evidence that they are insufficient for subsequent cortex areas *in vivo* [5]. We will show that our data recorded in prefrontal cortex suggest that higher order interactions (which involve more than two neurons) do play an important role in the prefrontal cortex as well.

In this paper, we present a method that addresses the above shortcomings of the multivariate normal distribution. We apply *copulas* [6] to form multivariate distributions with a rich set of dependence structures and discrete marginal distributions, including the Poisson distribution. Copulas were previously applied to model the distribution of continuous first-spike-latencies [7]. Here we apply this concept to spike counts.

## 2 Copulas

We give an informal introduction to copulas and apply the concept to a pair of neurons from our data which are described and fully analyzed in Section 4. Formal details of copulas follow in Section 3.2.

A copula is a cumulative distribution function that can couple arbitrary marginal distributions. There are many families of copulas, each with a different dependence structure. Some families have an elliptical dependence structure, similar to the multivariate normal distribution. However, it is also possible to use completely different dependence structures which are more appropriate for the data at hand.

As an example, consider the modeling of spike count dependencies of two neurons (Fig. 1). Spike trains are recorded from the neurons and transformed to spike counts (Fig. 1**a**). Counting leads to a bivariate empirical distribution (Fig. 1**b**). The distribution of the counts depends on the length of the time bin that is used to count the spikes, here 100 ms. In the case considered, the correlation at low counts is higher than at high counts. This is called *lower tail dependence*.

The density of a typical population model based on the multivariate normal (MVN) distribution is shown in Fig. 1**c**. Here, we did not discretize the distribution since the standard approach to investigate noise correlations also uses the continuous distribution [1]. The mean and covariance matrix of the MVN distribution correspond to the sample mean and the sample covariances of the empirical distribution. Yet, the dependence structure does not reflect the true dependence structure of the counts. But the spike count probabilities for a copula-based distribution (Fig. 1**d**) correspond well to the empirical distribution in Fig. 1**b**.

The modeling of spike count data with the help of a copula is done in three steps: (1) A marginal distribution, e.g. a Poisson or a negative binomial distribution is chosen, based on the spike count distribution of the individual neurons. (2) The counts are transformed to probabilities using the cumulative distribution function of the marginal distribution. (3) The probabilities and thereby the cumulative marginal distributions are coupled with the help of a so-called copula function. As an example, consider the Clayton copula family [6]. For two variables the copula is given by

$$C(p_1, p_2, \alpha) = \frac{1}{\sqrt[\alpha]{\max\{\frac{1}{p_1^\alpha} + \frac{1}{p_2^\alpha} - 1, \, 0\}}},$$

where $p_i$ denotes the probability of the spike count $X_i$ of the $i^{\text{th}}$ neuron being lower or equal to $r_i$ (i.e. $p_i = P(X_i \leq r_i)$). Note that there are generalizations to more than two margins (see Section 3.2). The function $C(p_1, p_2, \alpha)$ generates a joint cumulative distribution function by coupling the margins and thereby introduces correlations of second and higher order between the spike count variables. The ratio of the joint probability that corresponds to statistically independent spike counts $P(X_1 \leq r_1, X_2 \leq r_2) = p_1 p_2$ and the dependence introduced by the Clayton copula (for $\frac{1}{p_1^\alpha} + \frac{1}{p_2^\alpha} - 1 \geq 0$) is given by

$$\frac{p_1 p_2}{C(p_1, p_2, \alpha)} = p_1 p_2 \sqrt[\alpha]{\frac{1}{p_1^\alpha} + \frac{1}{p_2^\alpha} - 1} = \sqrt[\alpha]{p_1^\alpha + p_2^\alpha - p_1^\alpha p_2^\alpha}.$$

Suppose that $\alpha$ is positive. Since $p_i \in [0, 1]$ the deviation from the ratio 1 will be larger for small probabilities. Thus, the copula generates correlations whose strengths depend on the magnitude of the probabilities. The probability mass function (Fig. 1**d**) can then be calculated from the cumulative probability using the difference scheme as described in Section 3.4. Care must be taken whenever copulas are applied to form discrete distributions: while for continuous distributions typical measures of dependence are determined by the copula function $C$ only, these measures are affected by the shape of the marginal distributions in the discrete case [8].

## 3 Parametric spike count models and model selection procedure

We will now describe the formal aspects of the multivariate normal distribution on the one hand and copula-based models as the proposed alternative on the other hand, both in terms of their application to spike counts.

## 3.1 The discretized multivariate normal distribution

The MVN distribution is continuous and needs to be discretized (and rectified) before it can be applied to spike count data (which are discrete and non-negative). The cumulative distribution function (cdf) of the spike count vector $\vec{X}$ is then given by

$$F_{\vec{X}}(r_1, \ldots, r_d) = \begin{cases} \Phi_{\mu,\Sigma}(\lfloor r_1 \rfloor, \ldots, \lfloor r_d \rfloor), & \text{if } \forall i \in \{1, \ldots, d\} : r_i \geq 0 \\ 0, & \text{otherwise} \end{cases}$$

where $\lfloor . \rfloor$ denotes the floor operation for the discretization, $\Phi_{\mu,\Sigma}$ denotes the cdf of the MVN distribution with mean $\mu$ and correlation matrix $\Sigma$, and $d$ denotes the dimension of the multivariate distribution and corresponds to the number of neurons that are modeled. Note that $\mu$ is no longer the mean of $\vec{X}$. The mean is shifted to greater values as $\Phi_{\mu,\Sigma}$ is rectified (negative values are cut off). This deviation grows with the dimension $d$. According to the *central limit theorem*, the distribution of spike counts approaches the MVN distribution only for large counts.

## 3.2 Copula-based models

Formally, a copula $C$ is a cdf with uniform margins. It can be used to couple marginal cdf's $F_{X_1}, \ldots, F_{X_d}$ to form a joint cdf $F_{\vec{X}}$, such that

$$F_{\vec{X}}(r_1, \ldots, r_d) = C(F_{X_1}(r_1), \ldots, F_{X_d}(r_d))$$

holds [6]. There are many families of copulas with different dependence shapes and different numbers of parameters, e.g. the multivariate Clayton copula family with a scalar parameter $\alpha$:

$$C_\alpha(\vec{u}) = \left( \max \left\{ 1 - d + \sum_{i=1}^d u_i^{-\alpha}, 0 \right\} \right)^{-1/\alpha}.$$

Thus, for a given realization $\vec{r}$, which can represent the counts of two neurons, we can set $u_i = F_{X_i}(r_i)$ and $F_X(\vec{r}) = C_\alpha(\vec{u})$, where $F_{X_i}$ can be arbitrary univariate cdf's. Thereby, we can generate a multivariate distribution with specific margins $F_{X_i}$ and a dependence structure determined by $C$. In the case of discrete marginal distributions, however, typical measures of dependence, such as the linear correlation coefficient or Kendall's $\tau$ are effected by the shape of these margins [8]. Note that $\alpha$ does not only control the strength of pairwise interactions but also the degree of higher order interactions.

Another copula family is the Farlie-Gumbel-Morgenstern (FGM) copula [6]. It is special in that it has $2^d - d - 1$ parameters that individually determine the pairwise and higher order interactions. Its cdf takes the form

$$C_{\vec{\alpha}}(\vec{u}) = \left( 1 + \sum_{k=2}^d \sum_{1 \leq j_1 < \cdots < j_k \leq d} \alpha_{j_1 j_2 \ldots j_k} \prod_{i=1}^k (1 - u_{j_i}) \right) \prod_{i=1}^d u_i$$

subject to the constraints

$$1 + \sum_{k=2}^d \sum_{1 \leq j_1 < \cdots < j_k \leq d} \alpha_{j_1 j_2 \ldots j_k} \prod_{i=1}^k \varepsilon_{j_i} \geq 0, \quad \varepsilon_1, \varepsilon_2, \ldots \varepsilon_d \in \{-1, 1\}.$$

We only have pairwise interactions if we set all but the first $\binom{d}{2}$ parameters to zero. Hence, we can easily investigate the impact of higher order interactions on the model fit. Due to the constraints for $\alpha$, the correlations that the FGM copula can model are small in terms of their absolute value. Nevertheless, this is not an issue for modeling noise dependencies of spike counts of a small number of neurons, since the noise correlations that are found experimentally are typically small (see e.g. [2]).

## 3.3 Marginal distributions

Copulas allow us to have different marginal distributions. Typically, the Poisson distribution is a good approximation to spike count variations of single neurons [9]. For this distribution the cdf's of the margins take the form

$$F_{X_i}(r; \lambda_i) = \sum_{k=0}^{\lfloor r \rfloor} \frac{\lambda_i^k}{k!} e^{-\lambda_i},$$

where $\lambda_i$ is the mean spike count of neuron $i$ for a given bin size. We will also use the negative binomial distribution as a generalization of the Poisson distribution:

$$F_{X_i}(r; \lambda_i, \upsilon_i) = \sum_{k=0}^{\lfloor r \rfloor} \frac{\lambda_i^k}{k!} \frac{1}{(1 + \frac{\lambda_i}{\upsilon_i})^{\upsilon_i}} \frac{\Gamma(\upsilon_i + k)}{\Gamma(\upsilon_i)(\upsilon_i + \lambda_i)^k},$$

where $\Gamma$ is the gamma function. The additional parameter $\upsilon_i$ controls the degree of overdispersion: the smaller the value of $\upsilon_i$, the greater the Fano factor. As $\upsilon_i$ approaches infinity, the negative binomial distribution converges to the Poisson distribution.

## 3.4 Inference for copulas and discrete margins

Likelihoods of discrete vectors can be computed by applying the inclusion-exclusion principle of Poincaré and Sylvester. For this purpose we define the sets $A = \{X_1 \leq r_1, \ldots, X_d \leq r_d\}$ and $A_i = \{X_1 \leq r_1, \ldots, X_d \leq r_d, X_i \leq r_i - 1\}$, $i \in \{1, \ldots, d\}$. The probability of a realization $\vec{r}$ is given by

$$P_{\vec{X}}(\vec{r}) = P\left(A \setminus \bigcup_{i=1}^d A_i\right) = P(A) - \sum_{k=1}^d (-1)^{k-1} \sum_{\substack{I \subseteq \{1, \ldots, d\}, \\ |I| = k}} P\left(\bigcap_{i \in I} A_i\right)$$

$$= F_{\vec{X}}(\vec{r}) - \sum_{k=1}^d (-1)^{k-1} \sum_{\substack{\vec{m} \in \{0,1\}^d, \\ \sum m_i = k}} F_{\vec{X}}(r_1 - m_1, \ldots, r_d - m_d). \tag{1}$$

Thus, we can compute the probability mass of a realization $\vec{r}$ using only the cdf of $\vec{X}$. Since copulas separate the margins from the dependence structure, an efficient inference procedure is feasible. Let

$$l_i(\theta_i) = \sum_{t=1}^T \log P_{X_i}(r_{i,t}; \theta_i), \quad i = 1, \ldots, d$$

denote the univariate margins of log likelihoods. Note that we assume independent time bins. Further, let

$$l(\vec{\alpha}, \theta_1, \ldots, \theta_d) = \sum_{t=1}^T \log P_{\vec{X}}(\vec{r_t}; \vec{\alpha}, \theta_1, \ldots, \theta_d)$$

be the log likelihood of the joint distribution, where $\vec{\alpha}$ denotes the parameter of the copula. The so-called *inference for margins* (IFM) method proceeds in two steps [10]. First, the marginal likelihoods are maximized separately:

$$\widehat{\theta}_i = \underset{\theta_i}{\operatorname{argmax}}\{l_i(\theta_i)\}.$$

Then, the full likelihood is maximized given the estimated margin parameters:

$$\widehat{\vec{\alpha}} = \underset{\vec{\alpha}}{\operatorname{argmax}}\{l(\vec{\alpha}, \widehat{\theta}_1, \ldots, \widehat{\theta}_d)\}.$$

The estimator is asymptotically efficient and close to the maximum likelihood estimator [10].

## 3.5 Estimation of mutual information

The mutual information [11] of dependent spike counts $\vec{X}$ is a measure of the information that knowing the neural response $\vec{r}$ provides about the stimulus. It can be written as

$$I(\vec{X}; S) = \sum_{s \in M_S} P_S(s) \sum_{\vec{r} \in \mathbb{N}^d} P_{\vec{X}}(\vec{r}|s) \left( \log_2\left(P_{\vec{X}}(\vec{r}|s)\right) - \log_2\left(\sum_{s' \in M_S} P_S(s') P_{\vec{X}}(\vec{r}|s')\right) \right)$$

where $S$ is the stimulus random variable, $M_S$ is the set of stimuli, and $P_S$ is the probability mass function for the stimuli. The likelihood $P_{\vec{X}}(\vec{r}|s)$ of $\vec{r}$ given $s$ can be calculated using Equation 1. Thereby, $I(\vec{X}; S)$ can be estimated by the Monte Carlo method.

# 4    Application to multi-electrode recordings

We now apply our parametric count models to the analysis of spike data, which we recorded from the prefrontal cortex of an awake behaving macaque, using a $4 \times 4$ tetrode array.

**Experimental setup.**    Activity was recorded while the monkey performed a visual match-to-sample-task. The task involved matching of 20 visual stimuli (fruits and vegetables) that were presented for approximately 650 ms each. After an initial presentation ("sample") a test stimulus ("test") was presented with a delay of 3 seconds and the monkey had to decide by differential button press whether both stimuli were the same or not. Correct responses were rewarded. Match and non-match trials were randomly presented with an equal probability.

We recorded from the lateral prefrontal cortex in a $2 \times 2$ mm$^2$ area around the ventral bank of the principal sulcus. Recordings were performed simultaneously from up to 16 adjacent sites with an array of individually movable fiber micro-tetrodes (manufactured by Thomas Recording). Data were sampled at 32 kHz and bandpass filtered between 0.5 kHz and 10 kHz. Recording positions of individual electrodes were chosen to maximize the recorded activity and the signal quality.

The recorded data were processed by a PCA based spike sorting method. The method provides automatic cluster cutting which was manually corrected by subsequent cluster merging if indicated by quantitative criteria such as the ISI-histograms or amplitude stability.

**Data set.**    To select neurons with stimulus specific responses, we calculated spike counts from their spike trains. No neuron was accepted in the dependence analysis that shifted its mean firing rate averaged over the time interval of the sample stimulus presentation by less than 6.5 Hz compared to the pre-stimulus interval. A total of six neurons fulfilled this criterion (each recorded from a different tetrode). With this criterion we can assume that the selected neurons are indeed related to processing of the stimulus information.

Spike trains were separated into 80 groups, one for each of the 20 different stimuli and the four trial intervals: pre-stimulus, sample stimulus presentation, delay, and test stimulus presentation. Afterwards, the trains were binned into successive 100 ms intervals and converted to six-dimensional spike counts for each bin. Due to the different interval lengths, total sample sizes of the groups were between 224 and 1793 count vectors. A representative example of the empirical distribution of a pair of these counts from the stimulus presentation interval is presented in Fig. 1**b**.

**Model fitting.**    The discretized MVN distribution as well as several copula-based distributions were fitted to the data. For each of the 80 groups we selected randomly 50 count vectors (test set) for obtaining an unbiased estimate of the likelihoods. We trained the model on the remainder of each group (training set).

A commonly applied criterion for model selection is maximum entropy [4]. This criterion selects a certain model with minimal complexity subject to given constraints. It thereby performs regularization which is supposed to prevent overfitting. Copulas on the other hand typically increase the complexity of the model and thus decrease the entropy. However, our evaluation takes place on a separate test set and hence takes overfitting into account.

Parameter inference for the discretized MVN distribution (see Section 3.1) was performed by computing the sample mean and sample covariance matrix of the spike counts which is the standard procedure for analyzing noise correlations [1]. Note that this estimator is biased, since it is not the maximum likelihood solution for the discretized distribution.

The following copula families were used to construct noise distributions of the spike counts. The Clayton (see Section 3.2), Gumbel-Hougaard, Frank and Ali-Mikhail-Haq copula families as examples of families with one parameter [6] and the FGM with a variable number of parameters (see Section 3.2).

We applied the IFM method for copula inference (see Section 3.4). The sample mean is the maximum likelihood estimator for $\lambda_i$ for both the Poisson and the negative binomial margins. The maximum likelihood estimates for $\upsilon_i$ were computed iteratively by Newton's method. Depending on whether the copula parameters were constrained, either the Nelder-Mead simplex method for

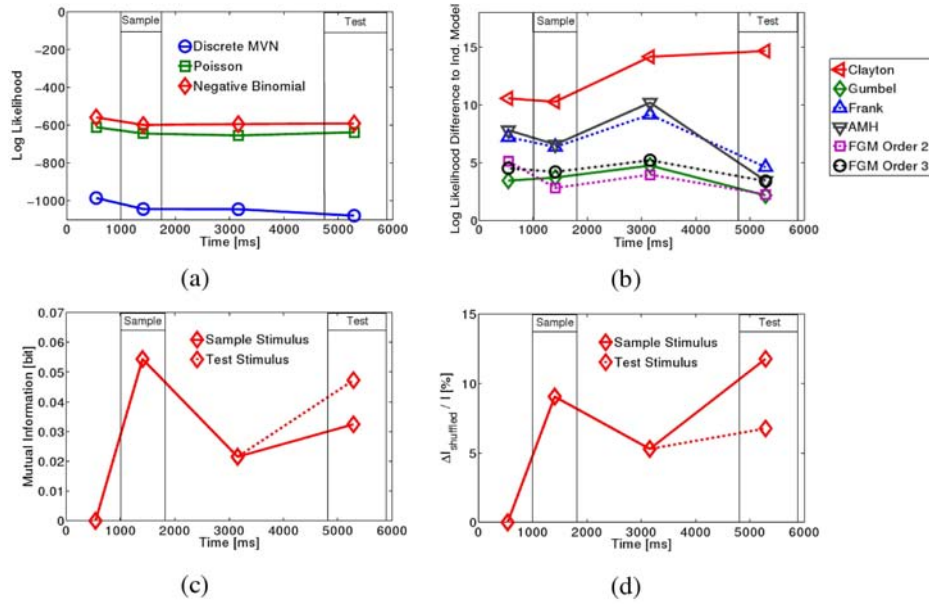

Figure 2: Evaluation of the IFM estimates on the test set and estimated mutual information. (**a**): Log likelihoods for the discrete multivariate normal distribution, the best fitting copula-based model with Poisson margins, and the best fitting copula-based model with negative binomial margins averaged over the 20 different stimuli. (**b**): Difference between the log likelihood of the model with independent counts and negative binomial margins ("ind. model") and the log likelihoods of different copula-based models with negative binomial margins averaged over the 20 different stimuli. (**c**): Mutual information between stimuli and responses for the Clayton-based model with negative binomial margins. (**d**): Normalized difference between the mutual information for the Clayton-based model with negative binomial margins and the corresponding "ind. model".

unconstrained nonlinear optimization or the line-search algorithm for constrained nonlinear optimization was applied to estimate the copula parameters.

**Results for different distributions.** Fig. 2 shows the evaluation of the IFM estimates on the test set. The likelihood for the copula-based models is significantly larger than for the discrete MVN model ($p = 2 \cdot 10^{-14}$, paired-sample Student's $t$ test over stimuli). Moreover, the likelihood for the negative binomial margins is even larger than that for the Poisson margins ($p = 0.0003$).

We estimated the impact of neglecting higher order interactions on the fit by using different numbers of parameters for the FGM copula. For the $2^{\text{nd}}$ order model we set all but the first $\binom{d}{2}$ parameters to zero, therefore leaving only parameters for pairwise interactions. In contrast, for the $3^{\text{rd}}$ order model we set all but the first $\binom{d}{2} + \binom{d}{3}$ parameters to zero.

We computed the difference between the likelihood of the model with dependence and the corresponding model with independence between its counts. Fig. 2**b** shows this difference for several copulas and negative binomial margins evaluated on the test set. The model based on the Clayton copula family provides the best fit. The fit is significantly better than for the second best fitting copula family ($p = 0.0014$). In spite of having more parameters, the FGM copulas perform worse. However, the FGM model with third order interactions fits the data significantly better than the model that includes only pairwise interactions ($p = 0.0437$).

**Copula coding analysis.** Fig. 2**c** shows the Monte Carlo estimate of the mutual information based on the Clayton-based model with negative binomial margins and IFM parameters determined on the training set for each of the intervals. For the test stimulus interval, the estimation was performed twice: for the previously presented sample stimulus and for the test stimulus. The Monte Carlo method was terminated when the standard error was below $5 \cdot 10^{-4}$. The mutual information is higher during the stimulus presentation intervals than during the delay interval.

We estimated the information increase due to the dependence structure by computing the mutual information for the Clayton-based model with negative binomial margins and subtracting the (smaller) mutual information for the corresponding distribution with independent elements. Fig. 2**d** shows this information estimate $\Delta I_{shuffled}$, normalized to the mutual information for the Clayton-based model. The dependece structure carries up to $12\%$ of the mutual information. During the test stimulus interval it carries almost twice as much information about the test stimulus as about the previously presented sample stimulus.

Another important measure related to stimulus decoding which is currently under debate is $\Delta I/I$ [12]. The measure provides an upper bound on the information loss for stimulus decoding based on the distribution that assumes independence. We find that one loses at most $19.82\%$ of the information for the Clayton-based model.

## 5 Conclusion

We developed a framework for analyzing the noise dependence of spike counts. Applying this to our data from the macaque prefrontal cortex we found that: (1) Gaussian noise is inadequate to model spike count data for short time intervals; (2) negative binomial distributed margins describe the individual spike counts better than Poisson distributed margins; and (3) higher order interactions are present and play a substantial role in terms of model fit and information content.

The substantial role of higher order interactions bears a challenge for theoreticians as well as experimentalists. The complexity of taking all higher order interactions into account grows exponentially with the number of neurons, known as the curse of dimensionality. Based on our findings, we conclude that one needs to deal with this problem to analyze short-term coding in higher cortical areas.

In summary, one can say that the copula-based approach provides a convenient way to study spike count dependencies for small population sizes ($< 20$). At present, the approach is computationally too demanding for higher numbers of neurons. Approximate inference methods might provide a solution to the computational problem and seem worthwhile to investigate. Directions for future research are the exploration of other copula families and the validation of population coding principles that were obtained on the assumption of Gaussian noise.

**Acknowledgments.** This work was supported by BMBF grant 01GQ0410.

## References

[1] B. B. Averbeck and P. E. Latham and A. P. Pouget, Neural correlations, population coding and computation. *Nature Review Neuroscience*, 7:358–366, 2006.

[2] W. Bair, E. Zohary, and W. T. Newsome, Correlated firing in macaque visual area MT: time scales and relationship to behavior. *Journal of Neuroscience*, 21(5):1676–1697, 2001.

[3] A. Kohn and M. A. Smith, Stimulus dependence of neuronal correlation in primary visual cortex of the macaque. *Journal of Neuroscience*, 25(14):3661–3673, 2005.

[4] E. Schneidman and M. J. Berry II and R. Segev and W. Bialek, Weak pairwise correlations imply strongly correlated network states in a neural population. *Nature*, 440:1007–1012, 2006.

[5] M. M. Michel and R. A. Jacobs, The costs of ignoring high-order correlations in populations of model neurons. *Neural Computation*, 18:660–682, 2006.

[6] R. B. Nelsen, *An Introduction to Copulas*. Springer, New York, second edition, 2006.

[7] R. L. Jenison and R. A. Reale, The shape of neural dependence. *Neural Computation*, 16:665–672, 2004.

[8] C. Genest and J. Neslehova, A primer on discrete copulas. *ASTIN Bulletin*, 37:475–515, 2007.

[9] D. J. Tolhurst, J. A. Movshon, and A. F. Dean, The statistical reliability of signals in single neurons in cat and monkey visual cortex. *Vision Research*, 23:775–785, 1982.

[10] H. Joe and J. J. Xu, The estimation method of inference functions for margins for multivariate models. *Technical Report*, 166, Department of Statistics, University of British Colombia, 1996.

[11] C. E. Shannon and W. Weaver, The mathematical theory of communication. Urbana: University of Illinois Press, 1949.

[12] P. E. Latham and S. Nirenberg, Synergy, redundancy, and independence in population codes, revisited. *Journal of Neuroscience*, 25(21):5195–5206, 2005.